# Discriminative Learning for Label Sequences via Boosting

**Yasemin Altun, Thomas Hofmann and Mark Johnson***
Department of Computer Science
*Department of Cognitive and Linguistics Sciences
Brown University, Providence, RI 02912
{altun,th}@cs.brown.edu, Mark_Johnson@brown.edu

## Abstract

This paper investigates a boosting approach to discriminative learning of label sequences based on a sequence rank loss function. The proposed method combines many of the advantages of boosting schemes with the efficiency of dynamic programming methods and is attractive both, conceptually and computationally. In addition, we also discuss alternative approaches based on the Hamming loss for label sequences. The sequence boosting algorithm offers an interesting alternative to methods based on HMMs and the more recently proposed Conditional Random Fields. Applications areas for the presented technique range from natural language processing and information extraction to computational biology. We include experiments on named entity recognition and part-of-speech tagging which demonstrate the validity and competitiveness of our approach.

## 1 Introduction

The problem of annotating or segmenting observation sequences arises in many applications across a variety of scientific disciplines, most prominently in natural language processing, speech recognition, and computational biology. Well-known applications include part-of-speech (POS) tagging, named entity classification, information extraction, text segmentation and phoneme classification in text and speech processing [7] as well as problems like protein homology detection, secondary structure prediction or gene classification in computational biology [3].

Up to now, the predominant formalism for modeling and predicting label sequences has been based on Hidden Markov Models (HMMs) and variations thereof. Yet, despite its success, generative probabilistic models – of which HMMs are a special case – have two major shortcomings, which this paper is not the first one to point out. First, generative probabilistic models are typically trained using maximum likelihood estimation (MLE) for a joint sampling model of observation and label sequences. As has been emphasized frequently, MLE based on the joint probability model is inherently *non-discriminative* and thus may lead to suboptimal prediction accuracy. Secondly, efficient inference and learning in this setting often requires

to make questionable conditional independence assumptions. More precisely, in the case of HMMs, it is assumed that the Markov blanket of the hidden label variable at time step $t$ consists of the previous and next labels as well as the $t$-th observation. This implies that all dependencies on past and future observations are mediated through neighboring labels.

In this paper, we investigate the use of discriminative learning methods for learning label sequences. This line of research continues previous approaches for learning conditional models, namely Conditional Random Fields (CRFs) [6], and discriminative re-ranking [1, 2]. CRFs have two main advantages compared to HMMs: They are trained discriminatively by maximizing a conditional (or pseudo-) likelihood criterion and they are more flexible in modeling additional dependencies such as direct dependencies of the $t$-th label on past or future observations. However, we strongly believe there are two further lines of research that are worth pursuing and may offer additional benefits or improvements.

First of all, and this is the main emphasis of this paper, an exponential loss function such as the one used in boosting algorithms [9, 4] may be preferable to the logarithmic loss function used in CRFs. In particular we will present a boosting algorithm that has the additional advantage of performing implicit feature selection, typically resulting in very sparse models. This is important for model regularization as well as for reasons of efficiency in high dimensional feature spaces. Secondly, we will also discuss the use of loss functions that explicitly minimize the zero/one loss on labels, i.e. the Hamming loss, as an alternative to loss functions based on ranking or predicting entire label sequences.

## 2   Additive Models and Exponential Families

Formally, learning label sequences is a generalization of the standard supervised classification problem. The goal is to learn a discriminant function for sequences, i.e. a mapping from observation sequences $\mathbf{X} = (\mathbf{x}_1, \mathbf{x}_2, \ldots, \mathbf{x}_t, \ldots)$ to label sequences $\mathbf{Y} = (y_1, y_2, \ldots, y_t, \ldots)$. The availability of a training set of labeled sequences $\mathcal{X} \equiv \{(\mathbf{X}^i, \mathbf{Y}^i) : i = 1, \ldots, n\}$ to learn this mapping from data is assumed.

In this paper, we focus on discriminant functions that can be written as additive models. The models under consideration take the following general form:

$$F_\theta(\mathbf{X}, \mathbf{Y}) = \sum_t F_\theta(\mathbf{X}, \mathbf{Y}; t), \quad \text{with} \quad F_\theta(\mathbf{X}, \mathbf{Y}; t) = \sum_k \theta_k f_k(\mathbf{X}, \mathbf{Y}; t) \tag{1}$$

Here $f_k$ denotes a (discrete) *feature* in the language of maximum entropy modeling, or a *weak learner* in the language of boosting. In the context of label sequences $f_k$ will typically be either of the form $f_k^{(1)}(\mathbf{x}_{t+s}, y_t)$ (with $s \in \{-1, 0, 1\}$) or $f_k^{(2)}(y_{t-1}, y_t)$. The first type of features will model dependencies between the observation sequence $\mathbf{X}$ and the $t$-th label in the sequence, while the second type will model inter-label dependencies between neighboring label variables. For ease of presentation, we will assume that all features are binary, i.e. each learner corresponds to an indicator function. A typical way of defining a set of weak learners is as follows:

$$f_k^{(1)}(\mathbf{x}_{t+s}, y_t) \equiv \delta(y_t, y(k))\chi_k(\mathbf{x}_{t+s}) \tag{2}$$

$$f_k^{(2)}(y_{t-1}, y_t) \equiv \delta(y_t, y(k))\delta(y_{t-1}, \bar{y}(k)). \tag{3}$$

where $\delta$ denotes the Kronecker-$\delta$ and $\chi_k$ is a binary feature function that extracts a feature from an observation pattern; $y(k)$ and $\bar{y}(k)$ refer to the label values for which the weak learner becomes "active".

There is a natural way to associate a conditional probability distribution over label sequences $\mathbf{Y}$ with an additive model $F_\theta$ by defining an exponential family for every fixed observation sequence $\mathbf{X}$

$$P_\theta(\mathbf{Y}|\mathbf{X}) \equiv \frac{\exp\left[F_\theta(\mathbf{X}, \mathbf{Y})\right]}{Z_\theta(\mathbf{X})}, \quad Z_\theta(\mathbf{X}) \equiv \sum_{\mathbf{Y}} \exp\left[F_\theta(\mathbf{X}, \mathbf{Y})\right]. \qquad (4)$$

This distribution is in exponential normal form and the parameters $\theta$ are also called *natural* or *canonical* parameters. By performing the sum over the sequence index $t$, we can see that the corresponding sufficient statistics are given by $S_k(\mathbf{X}, \mathbf{Y}) \equiv \sum_t f_k(\mathbf{X}, \mathbf{Y}; t)$. These sufficient statistics simply count the number of times the feature $f_k$ has been "active" along the labeled sequence $(\mathbf{X}, \mathbf{Y})$.

## 3 Logarithmic Loss and Conditional Random Fields

In CRFs, the log-loss of the model with parameters $\theta$ w.r.t. a set of sequences $\mathcal{X}$ is defined as the negative sum of the conditional probabilities of each training label sequence given the observation sequence,

$$\mathcal{H}^{\log}(\theta; \mathcal{X}) \equiv -\sum_i \log P_\theta(\mathbf{Y}^i|\mathbf{X}^i) = -\sum_i F_\theta(\mathbf{X}^i, \mathbf{Y}^i) + \log Z_\theta(\mathbf{X}^i). \qquad (5)$$

Although [6] has proposed a modification of improved iterative scaling for parameter estimation in CRFs, gradient-based methods such as conjugate gradient descent have often found to be more efficient for minimizing the convex loss function in Eq. (5) (cf. [8]). The gradient can be readily computed as

$$\nabla_\theta \mathcal{H}^{\log} = \sum_i \mathbf{E}\left[S(\mathbf{X}, \mathbf{Y})|\mathbf{X} = \mathbf{X}^i\right] - S(\mathbf{X}^i, \mathbf{Y}^i), \qquad (6)$$

where expectations are taken w.r.t. $P_\theta(\mathbf{Y}|\mathbf{X})$. The stationary equations then simply state that uniformly averaged over the training data, the observed sufficient statistics should match their conditional expectations. Computationally, the evaluation of $S(\mathbf{X}^i, \mathbf{Y}^i)$ is straightforward counting, while summing over all sequences $\mathbf{Y}$ to compute $\mathbf{E}\left[S(\mathbf{X}, \mathbf{Y})|\mathbf{X} = \mathbf{X}^i\right]$ can be performed using dynamic programming, since the dependency structure between labels is a simple chain.

## 4 Ranking Loss Functions for Label Sequences

As an alternative to logarithmic loss functions, we propose to minimize an upper bound on the ranking loss [9] adapted to label sequences. The ranking loss of a discriminant function $F_\theta$ w.r.t. a set of training sequences is defined as

$$\mathcal{H}^{\mathrm{rnk}}(\theta; \mathcal{X}) = \sum_i \sum_{\mathbf{Y} \neq \mathbf{Y}^i} \Theta(F_\theta(\mathbf{X}^i, \mathbf{Y}) - F_\theta(\mathbf{X}^i, \mathbf{Y}^i)), \Theta(x) \equiv \begin{cases} 0 & \text{for } x < 0 \\ 1 & \text{otherwise} \end{cases} \qquad (7)$$

which is simply the sum of the number of label sequences that are ranked higher than or equal to the true label sequence over all training sequences. It is straightforward to see (based on a term by term comparison) that an upper bound on the rank loss is given by the following exponential loss function

$$\mathcal{H}^{\exp}(\theta; \mathcal{X}) \equiv \sum_i \sum_{\mathbf{Y} \neq \mathbf{Y}^i} \exp\left[F_\theta(\mathbf{X}^i, \mathbf{Y}) - F_\theta(\mathbf{X}^i, \mathbf{Y}^i)\right] = \sum_i \left[\frac{1}{P_\theta(\mathbf{Y}^i|\mathbf{X}^i)} - 1\right]. (8)$$

Interestingly this simply leads to a loss function that uses the inverse conditional probability of the true label sequence, if we define this probability via the exponential form in Eq. (4). Notice that compared to [1], we include all sequences and not just the top $N$ list generated by some external mechanism. As we will show shortly, an explicit summation is possible because of the availability of dynamic programming formulation to compute sums over all sequences efficiently.

In order to derive gradient equations for the exponential loss we can simply make use of the elementary facts

$$\nabla_\theta(-\log P(\theta)) = -\frac{\nabla_\theta P(\theta)}{P(\theta)}, \quad \text{and } \nabla_\theta \frac{1}{P(\theta)} = -\frac{\nabla_\theta P(\theta)}{P(\theta)^2} = \frac{\nabla_\theta(-\log P(\theta))}{P(\theta)}. \quad (9)$$

Then it is easy to see that

$$\nabla_\theta \mathcal{H}^{\exp}(\theta; \mathcal{X}) = \sum_i \frac{\mathbf{E}\left[S(\mathbf{X}, \mathbf{Y})|\mathbf{X} = \mathbf{X}^i\right] - S(\mathbf{X}^i, \mathbf{Y}^i)}{P_\theta(\mathbf{Y}^i|\mathbf{X}^i)}. \quad (10)$$

The only difference between Eq. (6) and Eq. (10) is the non-uniform weighting of different sequences by their inverse probability, hence putting more emphasis on training label sequences that receive a small overall (conditional) probability.

## 5  Boosting Algorithm for Label Sequences

As an alternative to a simple gradient method, we now turn to the derivation of a boosting algorithm, following the boosting formulation presented in [9]. Let us introduce a relative weight (or distribution) $D(i, \mathbf{Y})$ for each label sequence $\mathbf{Y}$ w.r.t. a training instance $(\mathbf{X}^i, \mathbf{Y}^i)$, i.e. $\sum_i \sum_{\mathbf{Y}} D(i, \mathbf{Y}) = 1$,

$$D(i, \mathbf{Y}) \equiv \frac{\exp\left[F_\theta(\mathbf{X}^i, \mathbf{Y}) - F_\theta(\mathbf{X}^i, \mathbf{Y}^i)\right]}{\sum_{j,} \sum_{\mathbf{Y}' \neq \mathbf{Y}^j} \exp\left[F_\theta(\mathbf{X}^j, \mathbf{Y}') - F_\theta(\mathbf{X}^j, \mathbf{Y}^j)\right]}, \quad \text{for } \mathbf{Y} \neq \mathbf{Y}^i \quad (11)$$

$$= D(i)\frac{P_\theta(\mathbf{Y}|\mathbf{X}^i)}{1 - P_\theta(\mathbf{Y}^i|\mathbf{X}^i)}, \quad D(i) \equiv \frac{P_\theta(\mathbf{Y}^i|\mathbf{X}^i)^{-1} - 1}{\sum_j \left[P_\theta(\mathbf{Y}^j|\mathbf{X}^j)^{-1} - 1\right]} \quad (12)$$

In addition, we define $D(i, \mathbf{Y}^i) = 0$. Eq. (12) shows how we can split $D(i, \mathbf{Y})$ into a relative weight for each training instance, given by $D(i)$, and a relative weight of each sequence, given by the re-normalized conditional probability $P_\theta(\mathbf{Y}|\mathbf{X}^i)$. Notice that $D(i) \to 0$ as we approach the perfect prediction case of $P_\theta(\mathbf{Y}^i|\mathbf{X}^i) \to 1$.

We define a boosting algorithm which in each round aims at minimizing the partition function or weight normalization constant $Z_k$ w.r.t. a weak learner $f_k$ and a corresponding optimal parameter increment $\triangle\theta_k$

$$Z_k(\triangle\theta_k) \equiv \sum_i D(i) \sum_{\mathbf{Y} \neq \mathbf{Y}^i} \frac{P_\theta(\mathbf{Y}|\mathbf{X}^i)}{1 - P_\theta(\mathbf{Y}^i|\mathbf{X}^i)} \exp\left[\triangle\theta_k\left(S_k(\mathbf{X}^i, \mathbf{Y}) - S_k(\mathbf{X}^i, \mathbf{Y}^i)\right)\right] \quad (13)$$

$$= \sum_b \left(\sum_i D(i)P(b|\mathbf{X}^i; k)\right) \exp\left[b\triangle\theta_k\right], \quad (14)$$

where $P_\theta(b|\mathbf{X}^i; k) = \sum_{\mathbf{Y} \in \mathbf{Y}(b; \mathbf{X}^i)} P_\theta(\mathbf{Y}|\mathbf{X}^i)/(1 - P_\theta(\mathbf{Y}^i|\mathbf{X}^i))$ and $\mathbf{Y}(b; \mathbf{X}^i) \equiv \{\mathbf{Y} : \mathbf{Y} \neq \mathbf{Y}^i \wedge (S_k(\mathbf{X}^i, \mathbf{Y}) - S_k(\mathbf{X}^i, \mathbf{Y}^i)) = b\}$. This minimization problem is only tractable if the number of features is small, since a dynamic programming run with accumulators [6] for every feature seems to be required in order to compute

the probabilities $P_\theta(b|\mathbf{X}^i; k)$, i.e. the probability for the $k$-th feature to be active exactly $b$ times, conditioned on the observation sequence $\mathbf{X}^i$.

In cases, where this is intractable (and we assume this will be the case in most applications), one can instead minimize an upper bound on every $Z_k$. The general idea is to exploit the convexity of the exponential function and to bound

$$\exp[x] \le \exp[x^{\min}]\frac{x^{\max} - x}{x^{\max} - x^{\min}} + \exp[x^{\max}]\frac{x - x^{\min}}{x^{\max} - x^{\min}}, \tag{15}$$

which is valid for every $x \in [x^{\min}; x^{\max}]$.

We introduce the following shorthand notation $u_{ik}(\mathbf{Y}) \equiv S_k(\mathbf{X}^i, \mathbf{Y}) - S_k(\mathbf{X}^i, \mathbf{Y}^i)$, $u_{ik}^{\max} \equiv \max_{\mathbf{Y} \neq \mathbf{Y}^i} u_{ik}(\mathbf{Y})$, $u_k^{\max} = \max_i u_{ik}^{\max}$, $u_{ik}^{\min} \equiv \min_{\mathbf{Y} \neq \mathbf{Y}^i} u_{ik}(\mathbf{Y})$, $u_k^{\min} \equiv \min_i u_{ik}^{\min}$ and $\pi_i(\mathbf{Y}) \equiv P_\theta(\mathbf{Y}|\mathbf{X}^i)/(1 - P_\theta(\mathbf{Y}^i|\mathbf{X}^i))$ which allows us to rewrite

$$Z_k(\triangle\theta_k) = \sum_i D(i) \sum_{\mathbf{Y} \neq \mathbf{Y}^i} \pi_i(\mathbf{Y}) \exp\left[\triangle\theta_k u_{ik}(\mathbf{Y})\right] \tag{16}$$

$$\le \sum_i D(i) \sum_{\mathbf{Y} \neq \mathbf{Y}^i} \pi_i(\mathbf{Y}) \left[\frac{u_{ik}^{\max} - u_{ik}(\mathbf{Y})}{u_{ik}^{\max} - u_{ik}^{\min}} e^{\triangle\theta_k u_{ik}^{\min}} + \frac{u_{ik}(\mathbf{Y}) - u_{ik}^{\min}}{u_{ik}^{\max} - u_{ik}^{\min}} e^{\triangle\theta_k u_{ik}^{\max}}\right]$$

$$= \sum_i D(i) \left(r_{ik} e^{\triangle\theta_k u_{ik}^{\min}} + (1 - r_{ik}) e^{\triangle\theta_k u_{ik}^{\max}}\right), \text{ where} \tag{17}$$

$$r_{ik} \equiv \sum_{\mathbf{Y} \neq \mathbf{Y}^i} \pi_i(\mathbf{Y}) \frac{u_{ik}^{\max} - u_{ik}(\mathbf{Y})}{u_{ik}^{\max} - u_{ik}^{\min}} \tag{18}$$

By taking the second derivative w.r.t. $\triangle\theta_k$ it is easy to verify that this is a convex function in $\triangle\theta_k$ which can be minimized with a simple line search.

If one is willing to accept a looser bound, one can instead work with the interval $[u_k^{\min}; u_k^{\max}]$ which is the union of the intervals $[u_{ik}^{\min}; u_{ik}^{\max}]$ for every training sequence $i$ and obtain the upper bound

$$Z_k(\triangle\theta_k) \le r_k e^{\triangle\theta_k u_k^{\min}} + (1 - r_k) e^{\triangle\theta_k u_k^{\max}} \tag{19}$$

$$r_k \equiv \sum_i D(i) \sum_{\mathbf{Y} \neq \mathbf{Y}^i} \pi_i(\mathbf{Y}) \frac{u_k^{\max} - u_{ik}(\mathbf{Y})}{u_k^{\max} - u_k^{\min}} \tag{20}$$

Which can be solved analytically

$$\triangle\theta_k = \frac{1}{u_k^{\max} - u_k^{\min}} \log\left(\frac{-r_k u_k^{\min}}{(1 - r_k) u_k^{\max}}\right), \tag{21}$$

but will in general lead to more conservative step sizes.

The final boosting procedure picks at every round the feature for which the upper bound on $Z_k$ is minimal and then performs an update of $\theta_k \leftarrow \theta_k + \triangle\theta_k$. Of course, one might also use more elaborate techniques to find the optimal $\triangle\theta_k$, once $f_k$ has been selected, since the upper bound approximation may underestimate the optimal step sizes. It is important to see that the quantities involved ($r_{ik}$ and $r_k$, respectively) are simple expectations of sufficient statistics that can be computed for all features simultaneously with a single dynamic programming run per sequence.

## 6  Hamming Loss for Label Sequences

In many applications one is primarily interested in the label-by-label loss or *Hamming loss* [9]. Here we investigate how to train models by minimizing an upper

bound on the Hamming loss. The following logarithmic loss aims at maximizing the log-probability for each individual label and is given by

$$\mathcal{F}^{\log}(\theta; \mathcal{X}) \equiv -\sum_i \sum_t \log P_\theta(y_t^i | \mathbf{X}^i) = -\sum_i \sum_t \log \sum_{\mathbf{Y}: y_t = y_t^i} P_\theta(\mathbf{Y} | \mathbf{X}^i). \quad (22)$$

Again, focusing on gradient descent methods, the gradient is given by

$$\nabla_\theta \mathcal{F}^{\log} = -\sum_i \sum_t \mathbf{E}\left[ S(\mathbf{X}, \mathbf{Y}) | \mathbf{X} = \mathbf{X}^i, y_t = y_t^i \right] - \mathbf{E}\left[ S(\mathbf{X}, \mathbf{Y}) | \mathbf{X} = \mathbf{X}^i \right] \quad (23)$$

As can be seen, the expected sufficient statistics are now compared not to their empirical values, but to their expected values, conditioned on a given label value $y_t^i$ (and not the entire sequence $\mathbf{Y}^i$). In order to evaluate these expectations, one can perform dynamic programming using the algorithm described in [5], which has (independently of our work) focused on the use of Hamming loss functions in the context of CRFs. This algorithm has the complexity of the forward-backward algorithm scaled by a constant.

Similar to the log-loss case, one can define an exponential loss function that corresponds to a margin-like quantity at every single label. We propose minimizing the following loss function

$$\mathcal{F}^{\exp}(\theta; \mathcal{X}) = \sum_i \sum_t \sum_{\mathbf{Y}} \exp\left[ F_\theta(\mathbf{X}^i, \mathbf{Y}) - \log \sum_{\mathbf{Y}', y_t' = y_t^i} \exp\left[ F_\theta(\mathbf{X}^i, \mathbf{Y}') \right] \right] \quad (24)$$

$$= \sum_{i,t} \frac{\sum_{\mathbf{Y}} \exp\left[ F_\theta(\mathbf{X}^i, \mathbf{Y}) \right]}{\sum_{\mathbf{Y}, y_t = y_t^i} \exp\left[ F_\theta(\mathbf{X}^i, \mathbf{Y}) \right]} = \sum_{i,t} P_\theta(y_t^i | \mathbf{X}^i; \theta)^{-1} \quad (25)$$

As a motivation, we point out that for the case of sequences of length 1, this will reduce to the standard multi-class exponential loss. Effectively in this model, the prediction of a label $y_t$ will mimic the probabilistic marginalization, i.e. $\hat{y}_t^i = \arg\max_y F_\theta(\mathbf{X}^i, y; t)$, $F_\theta(\mathbf{X}^i, y; t) = \log \sum_{\mathbf{Y}: y_t = y} \exp\left[ F_\theta(\mathbf{X}^i, \mathbf{Y}) \right]$.

Similar to the log-loss case, the gradient is given by

$$\nabla_\theta \mathcal{F}^{\exp} = -\sum_{i,t} \frac{\mathbf{E}\left[ S(\mathbf{X}, \mathbf{Y}) | \mathbf{X} = \mathbf{X}^i, y_t = y_t^i \right] - \mathbf{E}\left[ S(\mathbf{X}^i, \mathbf{Y}) | \mathbf{X} = \mathbf{X}^i \right]}{P_\theta(y_t^i | \mathbf{X}^i)} \quad (26)$$

Again, we see the same differences between the log-loss and the exponential loss, but this time for individual labels. Labels for which the marginal probability $P_\theta(y_i^t | \mathbf{X}^i)$ is small are accentuated in the exponential loss. The computational complexity for computing $\nabla_\theta \mathcal{F}^{\exp}$ and $\nabla_\theta \mathcal{F}^{\log}$ is practically the same. We have not been able to derive a boosting formulation for this loss function, mainly because it cannot be written as a sum of exponential terms. We have thus resorted to conjugate gradient descent methods for minimizing $\mathcal{F}^{\exp}$ in our experiments.

# 7 Experimental Results

## 7.1 Named Entity Recognition

Named Entity Recognition (NER), a subtask of Information Extraction, is the task of finding the phrases that contain person, location and organization names, times and quantities. Each word is tagged with the *type* of the name as well as its *position* in the name phrase (i.e. whether it is the first item of the phrase or not) in order to represent the boundary information.

We used a Spanish corpus which was provided for the Special Session of CoNLL2002 on NER. The data is a collection of news wire articles and is tagged for person names, organizations, locations and miscellaneous names.

We used simple binary features to ask questions about the word being tagged, as well as the previous tag (i.e. HMM features). An example feature would be: *Is the current word='Clinton' and the tag='Person-Beginning'?*. We also used features to ask detailed questions (i.e. spelling features) about the current word (e.g.: *Is the current word capitalized and the tag='Location-Intermediate'?*) and the neighboring words. These questions cannot be asked (in a principled way) in a generative HMM model. We ran experiments comparing the different loss functions optimized with the conjugate gradient method and the boosting algorithm. We designed three sets of features: HMM features (=$S1$), $S1$ and detailed features of the current word (=$S2$), and $S2$ and detailed features of the neighboring words (=$S3$).

The results summarized in Table 1 demonstrate the competitiveness of the proposed loss functions with respect to $\mathcal{H}^{\log}$. We observe that with different sets of features, the ordering of the performance of the loss functions changes. Boosting performs worse than the conjugate gradient when only HMM features are used, since there is not much information in the features other than the identity of the word to be labeled. Consequently, the boosting algorithm needs to include almost all weak learners in the ensemble and cannot exploit feature sparseness. When there are more detailed features, the boosting algorithm is competitive with the conjugate gradient method, but has the advantage of generating sparser models. The conjugate gradient method uses all of the available features, whereas boosting uses only about 10% of the features.

| Feature Set | | Objective | | |
|---|---|---|---|---|
| | | log | exp | boost |
| S1 | $\mathcal{H}$ | **6.60** | 6.95 | 8.05 |
| | $\mathcal{F}$ | 6.73 | 7.33 | - |
| S2 | $\mathcal{H}$ | 6.72 | 7.03 | 6.93 |
| | $\mathcal{F}$ | **6.67** | 7.49 | - |
| S3 | $\mathcal{H}$ | 6.15 | 5.84 | 6.77 |
| | $\mathcal{F}$ | 5.90 | **5.10** | - |

Table 1: Test error of the Spanish corpus for named entity recognition.

## 7.2 Part of Speech Tagging

We used the Penn TreeBank corpus for the part-of-speech tagging experiments. The features were similar to the feature sets S1 and S2 described above in the context of NER. Table 2 summarizes the experimental results obtained on this task. It can be seen that the test errors obtained by different loss functions lie within a relatively small range. Qualitatively the behavior of the different optimization methods is comparable to the NER experiments.

| Feature Set | | Objective | | |
|---|---|---|---|---|
| | | log | exp | boost |
| S1 | $\mathcal{H}$ | **4.69** | 5.04 | 10.58 |
| | $\mathcal{F}$ | 4.88 | 4.96 | - |
| S2 | $\mathcal{H}$ | **4.37** | 4.74 | 5.09 |
| | $\mathcal{F}$ | 4.71 | 4.90 | - |

Table 2: Test error of the Penn TreeBank corpus for POS

## 7.3 General Comments

Even with the tighter bound in the boosting formulation, the same features are selected many times, because of the conservative estimate of the step size for parameter updates. We expect to speed up the convergence of the boosting algorithm

by using a more sophisticated line search mechanism to compute the optimal step length, a conjecture that will be addressed in future work.

Although we did not use real-valued features in our experiments, we observed that including real-valued features in a conjugate gradient formulation is a challenge, whereas it is very natural to have such features in a boosting algorithm.

We noticed in our experiments that defining a distribution over the training instances using the inverse conditional probability creates problems in the boosting formulation for data sets that are highly unbalanced in terms of the length of the training sequences. To overcome this problem, we divided the sentences into pieces such that the variation in the length of the sentences is small. The conjugate gradient optimization, on the other hand, did not appear to suffer from this problem.

## 8  Conclusion and Future Work

This paper makes two contributions to the problem of learning label sequences. First, we have presented an efficient algorithm for discriminative learning of label sequences that combines boosting with dynamic programming. The algorithm compares favorably with the best previous approach, Conditional Random Fields, and offers additional benefits such as model sparseness. Secondly, we have discussed the use of methods that optimize a label-by-label loss and have shown that these methods bear promise for further improving classification accuracy. Our future work will investigate the performance (in both accuracy and computational expenses) of the different loss functions in different conditions (e.g. noise level, size of the feature set).

### Acknowledgments

This work was sponsored by an NSF-ITR grant, award number IIS-0085940.

## References

[1] M. Collins. Discriminative reranking for natural language parsing. In *Proceedings 17th International Conference on Machine Learning*, pages 175–182. Morgan Kaufmann, San Francisco, CA, 2000.

[2] M. Collins. Ranking algorithms for named–entity extraction: Boosting and the voted perceptron. In *Proceedings 40th Annual Meeting of the Association for Computational Linguistics (ACL)*, pages 489–496, 2002.

[3] R. Durbin, S. Eddy, A. Krogh, and G. Mitchison. *Biological Sequence Analysis: Probabilistic Models of Proteins and Nucleic Acids*. Cambridge University Press, 1998.

[4] J. Friedman, T. Hastie, and R. Tibshirani. Additive logistic regression: a statistical view of boosting. *Annals of Statistics*, 28:337–374, 2000.

[5] S. Kakade, Y.W. Teh, and S. Roweis. An alternative objective function for Markovian fields. In *Proceedings 19th International Conference on Machine Learning*, 2002.

[6] J. Lafferty, A. McCallum, and F. Pereira. Conditional random fields: Probabilistic models for segmenting and labeling sequence data. In *Proc. 18th International Conf. on Machine Learning*, pages 282–289. Morgan Kaufmann, San Francisco, CA, 2001.

[7] C. Manning and H. Schütze. *Foundations of Statistical Natural Language Processing*. MIT Press, 1999.

[8] T. Minka. Algorithms for maximum-likelihood logistic regression. Technical report, CMU, Department of Statistics, TR 758, 2001.

[9] R. Schapire and Y. Singer. Improved boosting algorithms using confidence-rated predictions. *Machine Learning*, 37(3):297–336, 1999.
